# A Unifying Perspective of Parametric Policy Search Methods for Markov Decision Processes

**Thomas Furmston**
Department of Computer Science
University College London
T.Furmston@cs.ucl.ac.uk

**David Barber**
Department of Computer Science
University College London
D.Barber@cs.ucl.ac.uk

## Abstract

Parametric policy search algorithms are one of the methods of choice for the optimisation of Markov Decision Processes, with Expectation Maximisation and natural gradient ascent being popular methods in this field. In this article we provide a unifying perspective of these two algorithms by showing that their search-directions in the parameter space are closely related to the search-direction of an approximate Newton method. This analysis leads naturally to the consideration of this approximate Newton method as an alternative optimisation method for Markov Decision Processes. We are able to show that the algorithm has numerous desirable properties, absent in the naive application of Newton's method, that make it a viable alternative to either Expectation Maximisation or natural gradient ascent. Empirical results suggest that the algorithm has excellent convergence and robustness properties, performing strongly in comparison to both Expectation Maximisation and natural gradient ascent.

## 1 Markov Decision Processes

Markov Decision Processes (MDPs) are the most commonly used model for the description of sequential decision making processes in a fully observable environment, see *e.g.* [5]. A MDP is described by the tuple $\{\mathcal{S}, \mathcal{A}, H, p_1, p, \pi, R\}$, where $\mathcal{S}$ and $\mathcal{A}$ are sets known respectively as the state and action space, $H \in \mathbb{N}$ is the planning horizon, which can be either finite or infinite, and $\{p_1, p, \pi, R\}$ are functions that are referred as the *initial state distribution*, *transition dynamics*, *policy* (or *controller*) and the *reward function*. In general the state and action spaces can be arbitrary sets, but we restrict our attention to either discrete sets or subsets of $\mathbb{R}^n$, where $n \in \mathbb{N}$. We use boldface notation to represent a vector and also use the notation $\boldsymbol{z} = (\boldsymbol{s}, \boldsymbol{a})$ to denote a state-action pair. Given a MDP the trajectory of the agent is determined by the following recursive procedure: Given the agent's state, $\boldsymbol{s}_t$, at a given time-point, $t \in \mathbb{N}_H$, an action is selected according to the policy, $\boldsymbol{a}_t \sim \pi(\cdot|\boldsymbol{s}_t)$; The agent will then transition to a new state according to the transition dynamics, $\boldsymbol{s}_{t+1} \sim p(\cdot|\boldsymbol{a}_t, \boldsymbol{s}_t)$; this process is iterated sequentially through all of the time-points in the planning horizon, where the state of the initial time-point is determined by the initial state distribution $\boldsymbol{s}_1 \sim p_1(\cdot)$. At each time-point the agent receives a (scalar) reward that is determined by the reward function, where this function depends on the current action and state of the environment. Typically the reward function is assumed to be bounded, but as the objective is linear in the reward function we assume *w.l.o.g* that it is non-negative.

The most widely used objective in the MDP framework is to maximise the total expected reward of the agent over the course of the planning horizon. This objective can take various forms, including an infinite planning horizon, with either discounted or average rewards, or a finite planning horizon. The theoretical contributions of this paper are applicable to all three frameworks, but for notational ease and for reasons of space we concern ourselves with the infinite horizon framework with discounted rewards. In this framework the boundedness of the objective function is ensured by the

introduction of a discount factor, $\gamma \in [0, 1)$, which scales the rewards of the various time-points in a geometric manner. Writing the objective function and trajectory distribution directly in terms of the parameter vector then, for any $\boldsymbol{w} \in \mathcal{W}$, the objective function takes the form

$$U(\boldsymbol{w}) = \sum_{t=1}^{\infty} \mathbb{E}_{p_t(\boldsymbol{a}, \boldsymbol{s}; \boldsymbol{w})}\left[\gamma^{t-1} R(\boldsymbol{a}, \boldsymbol{s})\right], \tag{1}$$

where we have denoted the parameter space by $\mathcal{W}$ and have used the notation $p_t(\boldsymbol{a}, \boldsymbol{s}; \boldsymbol{w})$ to represent the marginal $p(\boldsymbol{s}_t = \boldsymbol{s}, \boldsymbol{a}_t = \boldsymbol{a}; \boldsymbol{w})$ of the joint state-action trajectory distribution

$$p(\boldsymbol{a}_{1:H}, \boldsymbol{s}_{1:H}; \boldsymbol{w}) = \pi(\boldsymbol{a}_H | \boldsymbol{s}_H; \boldsymbol{w})\left\{ \prod_{t=1}^{H-1} p(\boldsymbol{s}_{t+1} | \boldsymbol{a}_t, \boldsymbol{s}_t) \pi(\boldsymbol{a}_t | \boldsymbol{s}_t; \boldsymbol{w}) \right\} p_1(\boldsymbol{s}_1), \quad H \in \mathbb{N}. \tag{2}$$

Note that the policy is now written in terms of its parametric representation, $\pi(\boldsymbol{a}|\boldsymbol{s}; \boldsymbol{w})$.

It is well known that the global optimum of (1) can be obtained through dynamic programming, see *e.g.* [5]. However, due to various issues, such as prohibitively large state-action spaces or highly non-linear transition dynamics, it is not possible to find the global optimum of (1) in most real-world problems of interest. Instead most research in this area focuses on obtaining approximate solutions, for which there exist numerous techniques, such as approximate dynamic programming methods [6], Monte-Carlo tree search methods [19] and policy search methods, both parametric [27, 21, 16, 18] and non-parametric [2, 25].

This work is focused solely on parametric policy search methods, by which we mean gradient-based methods, such as steepest and natural gradient ascent [23, 1], along with Expectation Maximisation [11], which is a bound optimisation technique from the statistics literature. Since their introduction [14, 31, 10, 16] these methods have been the centre of a large amount of research, with much of it focusing on gradient estimation [21, 4], variance reduction techniques [30, 15], function approximation techniques [27, 8, 20] and real-world applications [18, 26]. While steepest gradient ascent has enjoyed some success it is known to suffer from some substantial issues that often make it unattractive in practice, such as slow convergence and susceptibility to poor scaling of the objective function [23]. Various optimisation methods have been introduced as an alternative, most notably natural gradient ascent [16, 24, 3] and Expectation Maximisation [18, 28], which are currently the methods of choice among parametric policy search algorithms. In this paper our primary focus is on the search-direction (in the parameter space) of these two methods.

## 2   Search Direction Analysis

In this section we will perform a novel analysis of the search-direction of both natural gradient ascent and Expectation Maximisation. In gradient-based algorithms of Markov Decision Processes the update of the policy parameters take the form

$$\boldsymbol{w}^{\text{new}} = \boldsymbol{w} + \alpha \mathcal{M}(\boldsymbol{w}) \nabla_{\boldsymbol{w}} U(\boldsymbol{w}), \tag{3}$$

where $\alpha \in \mathbb{R}^+$ is the step-size parameter and $\mathcal{M}(\boldsymbol{w})$ is some positive-definite matrix that possibly depends on $\boldsymbol{w}$. It is well-known that such an update will increase the total expected reward, provided that $\alpha$ is sufficiently small, and this process will converge to a local optimum of (1) provided the step-size sequence is appropriately selected. While EM doesn't have an update of the form given in (3) we shall see that the algorithm is closely related to such an update. It is convenient for later reference to note that the gradient $\nabla_{\boldsymbol{w}} U(\boldsymbol{w})$ can be written in the following form

$$\nabla_{\boldsymbol{w}} U(\boldsymbol{w}) = \mathbb{E}_{p_{\gamma}(\boldsymbol{z}; \boldsymbol{w}) Q(\boldsymbol{z}; \boldsymbol{w})}\left[\nabla_{\boldsymbol{w}} \log \pi(\boldsymbol{a}|\boldsymbol{s}; \boldsymbol{w})\right], \tag{4}$$

where we use the expectation notation $\mathbb{E}[\cdot]$ to denote the integral/summation *w.r.t.* a non-negative function. The term $p_{\gamma}(\boldsymbol{z}; \boldsymbol{w})$ is a geometric weighted average of state-action occupancy marginals given by

$$p_{\gamma}(\boldsymbol{z}; \boldsymbol{w}) = \sum_{t=1}^{\infty} \gamma^{t-1} p_t(\boldsymbol{z}; \boldsymbol{w}),$$

while the term $Q(\boldsymbol{z}; \boldsymbol{w})$ is referred to as the *state-action value function* and is equal to the total expected future reward from the current time-point onwards, given the current state-action pair, $\boldsymbol{z}$,

and parameter vector, $\boldsymbol{w}$, *i.e.*

$$Q(\boldsymbol{z}; \boldsymbol{w}) = \sum_{t=1}^{\infty} \mathbb{E}_{p_t(\boldsymbol{z}'; \boldsymbol{w})}\left[\gamma^{t-1}R(\boldsymbol{z}')\Big|\boldsymbol{z}_1 = \boldsymbol{z}\right].$$

This is a standard result and due to reasons of space we have omitted the details, but see *e.g.* [27] or section(6.1) of the supplementary material for more details.

An immediate issue concerning updates of the form (3) is in the selection of the 'optimal' choice of the matrix $\mathcal{M}(\boldsymbol{w})$, which clearly depends on the sense in which optimality is defined. There are numerous reasonable properties that are desirable of such an update, including the numerical stability and computational complexity of the parameter update, as well as the rate of convergence of the overall algorithm resulting from these updates. While all reasonable criteria the rate of convergence is of such importance in an optimisation algorithm that it is a logical starting point in our analysis. For this reason we concern ourselves with relating these two parametric policy search algorithms to the Newton method, which has the highly desirable property of having a quadratic rate of convergence in the vicinity of a local optimum. The Newton method is well-known to suffer from problems that make it either infeasible or unattractive in practice, but in terms of forming a basis for theoretical comparisons it is a logical starting point. We shall discuss some of the issues with the Newton method in more detail in section(3). In the Newton method the matrix $\mathcal{M}(\boldsymbol{w})$ is set to the negative inverse Hessian, *i.e.*

$$\mathcal{M}(\boldsymbol{w}) = -\mathcal{H}^{-1}(\boldsymbol{w}), \quad \text{where } \mathcal{H}(\boldsymbol{w}) = \nabla_{\boldsymbol{w}}\nabla_{\boldsymbol{w}}^T U(\boldsymbol{w}).$$

where we have denoted the Hessian by $\mathcal{H}(\boldsymbol{w})$. Using methods similar to those used to calculate the gradient, it can be shown that the Hessian takes the form

$$\mathcal{H}(\boldsymbol{w}) = \mathcal{H}_1(\boldsymbol{w}) + \mathcal{H}_2(\boldsymbol{w}), \tag{5}$$

where

$$\mathcal{H}_1(\boldsymbol{w}) = \sum_{t=1}^{\infty} \mathbb{E}_{p(\boldsymbol{z}_{1:t}; \boldsymbol{w})}\left[\gamma^{t-1}R(\boldsymbol{z}_t)\nabla_{\boldsymbol{w}}\log p(\boldsymbol{z}_{1:t}; \boldsymbol{w})\nabla_{\boldsymbol{w}}^T\log p(\boldsymbol{z}_{1:t}; \boldsymbol{w})\right], \tag{6}$$

$$\mathcal{H}_2(\boldsymbol{w}) = \sum_{t=1}^{\infty} \mathbb{E}_{p(\boldsymbol{z}_{1:t}; \boldsymbol{w})}\left[\gamma^{t-1}R(\boldsymbol{z}_t)\nabla_{\boldsymbol{w}}\nabla_{\boldsymbol{w}}^T\log p(\boldsymbol{z}_{1:t}; \boldsymbol{w})\right]. \tag{7}$$

We have omitted the details of the derivation, but these can be found in section(6.2) of the supplementary material, with a similar derivation of a sample-based estimate of the Hessian given in [4].

## 2.1 Natural Gradient Ascent

To overcome some of the issues that can hinder steepest gradient ascent an alternative, *natural gradient*, was introduced in [16]. Natural gradient ascent techniques originated in the neural network and blind source separation literature, see *e.g.* [1], and take the perspective that the parameter space has a Riemannian manifold structure, as opposed to a Euclidean structure. Deriving the steepest ascent direction of $U(\boldsymbol{w})$ *w.r.t.* a local norm defined on this parameter manifold (as opposed to *w.r.t.* the Euclidean norm, which is the case in steepest gradient ascent) results in natural gradient ascent. We denote the quadratic form that induces this local norm on the parameter manifold by $G(\boldsymbol{w})$, *i.e.* $d(\boldsymbol{w})^2 = \boldsymbol{w}^T G(\boldsymbol{w})\boldsymbol{w}$. The derivation for natural gradient ascent is well-known, see *e.g.* [1], and its application to the objective (1) results in a parameter update of the form

$$\boldsymbol{w}_{k+1} = \boldsymbol{w}_k + \alpha_k G^{-1}(\boldsymbol{w}_k)\nabla_{\boldsymbol{w}}U(\boldsymbol{w}_k).$$

In terms of (3) this corresponds to $\mathcal{M}(\boldsymbol{w}) = G^{-1}(\boldsymbol{w})$. In the case of MDPs the most commonly used local norm is given by the Fisher information matrix of the trajectory distribution, see *e.g.* [3, 24], and due to the Markovian structure of the dynamics it is given by

$$G(\boldsymbol{w}) = -\mathbb{E}_{p_\gamma(\boldsymbol{z}; \boldsymbol{w})}\left[\nabla_{\boldsymbol{w}}\nabla_{\boldsymbol{w}}^T\log\pi(\boldsymbol{a}|\boldsymbol{s}; \boldsymbol{w})\right]. \tag{8}$$

We note that there is an alternate, but equivalent, form of writing the Fisher information matrix, see *e.g.* [24], but we do not use it in this work.

In order to relate natural gradient ascent to the Newton method we first rewrite the matrix (7) into the following form

$$\mathcal{H}_2(\boldsymbol{w}) = \mathbb{E}_{p_\gamma(\boldsymbol{z};\boldsymbol{w})Q(\boldsymbol{z};\boldsymbol{w})} \left[ \nabla_{\boldsymbol{w}} \nabla_{\boldsymbol{w}}^T \log \pi(\boldsymbol{a}|\boldsymbol{s};\boldsymbol{w}) \right]. \tag{9}$$

For reasons of space the details of this reformulation of (7) are left to section(6.2) of the supplementary material. Comparing the Fisher information matrix (8) with the form of $\mathcal{H}_2(\boldsymbol{w})$ given in (9) it is clear that natural gradient ascent has a relationship with the approximate Newton method that uses $\mathcal{H}_2(\boldsymbol{w})$ in place of $\mathcal{H}(\boldsymbol{w})$. In terms of (3) this approximate Newton method corresponds to setting $\mathcal{M}(\boldsymbol{w}) = -\mathcal{H}_2^{-1}(\boldsymbol{w})$. In particular it can be seen that the difference between the two methods lies in the non-negative function *w.r.t.* which the expectation is taken in (8) and (9). (It also appears that there is a difference in sign, but observing the form of $\mathcal{M}(\boldsymbol{w})$ for each algorithm shows that this is not the case.) In the Fisher information matrix the expectation is taken *w.r.t.* to the geometrically weighted summation of state-action occupancy marginals of the trajectory distribution, while in $\mathcal{H}_2(\boldsymbol{w})$ there is an additional weighting from the state-action value function. Hence, $\mathcal{H}_2(\boldsymbol{w})$ incorporates information about the reward structure of the objective function, whereas the Fisher information matrix does not, and so it will generally contain more information about the curvature of the objective function.

## 2.2 Expectation Maximisation

The Expectation Maximisation algorithm, or EM-algorithm, is a powerful optimisation technique from the statistics literature, see *e.g.* [11], that has recently been the centre of much research in the planning and reinforcement learning communities, see *e.g.* [10, 28, 18]. A quantity of central importance in the EM-algorithm for MDPs is the following lower-bound on the $\log$-objective

$$\log U(\boldsymbol{w}) \geq \mathcal{H}_{\text{entropy}}(q(\boldsymbol{z}_{1:t}, t)) + \mathbb{E}_{q(\boldsymbol{z}_{1:t}, t)} \left[ \log \gamma^{t-1} R(\boldsymbol{z}_t) p(\boldsymbol{z}_{1:t}; \boldsymbol{w}) \right], \tag{10}$$

where $\mathcal{H}_{\text{entropy}}$ is the entropy function and $q(\boldsymbol{z}_{1:t}, t)$ is known as the 'variational distribution'. Further details of the EM-algorithm for MDPs and a derivation of (10) are given in section(6.3) of the supplementary material or can be found in *e.g.* [18, 28]. The parameter update of the EM-algorithm is given by the maximum (*w.r.t.* $\boldsymbol{w}$) of the 'energy' term,

$$\mathcal{Q}(\boldsymbol{w}, \boldsymbol{w}_k) = \mathbb{E}_{p_\gamma(\boldsymbol{z};\boldsymbol{w}_k)Q(\boldsymbol{z};\boldsymbol{w}_k)} \left[ \log \pi(\boldsymbol{a}|\boldsymbol{s};\boldsymbol{w}) \right]. \tag{11}$$

Note that $\mathcal{Q}$ is a two-parameter function, where the first parameter occurs inside the expectation and the second parameter defines the non-negative function *w.r.t.* the expectation is taken. This decoupling allows the maximisation over $\boldsymbol{w}$ to be performed explicitly in many cases of interest. For example, when the $\log$-policy is quadratic in $\boldsymbol{w}$ the maximisation problems is equivalent to a least-squares problem and the optimum can be found through solving a linear system of equations.

It has previously been noted, again see *e.g.* [18], that the parameter update of steepest gradient ascent and the EM-algorithm can be related through this 'energy' term. In particular the gradient (4) evaluated at $\boldsymbol{w}_k$ can also be written as follows $\nabla_{\boldsymbol{w}|\boldsymbol{w}=\boldsymbol{w}_k} U(\boldsymbol{w}) = \nabla_{\boldsymbol{w}|\boldsymbol{w}=\boldsymbol{w}_k}^{10} \mathcal{Q}(\boldsymbol{w}, \boldsymbol{w}_k)$, where we use the notation $\nabla_{\boldsymbol{w}}^{10}$ to denote the first derivative *w.r.t.* the first parameter, while the update of the EM-algorithm is given by $\boldsymbol{w}_{k+1} = \text{argmax}_{\boldsymbol{w} \in \mathcal{W}} \mathcal{Q}(\boldsymbol{w}, \boldsymbol{w}_k)$. In other words, steepest gradient ascent moves in the direction that most rapidly increases $\mathcal{Q}(\boldsymbol{w}, \boldsymbol{w}_k)$ *w.r.t.* the first variable, while the EM-algorithm maximises $\mathcal{Q}(\boldsymbol{w}, \boldsymbol{w}_k)$ *w.r.t.* the first variable. While this relationship is true, it is also quite a negative result. It states that in situations where it is not possible to explicitly perform the maximisation over $\boldsymbol{w}$ in (11) then the alternative, in terms of the EM-algorithm, is this generalised EM-algorithm, which is equivalent to steepest gradient ascent. Considering that algorithms such as EM are typically considered because of the negative aspects related to steepest gradient ascent this is an undesirable alternative. It is possible to find the optimum of (11) numerically, but this is also undesirable as it results in a double-loop algorithm that could be computationally expensive. Finally, this result provides no insight into the behaviour of the EM-algorithm, in terms of the direction of its parameter update, when the maximisation over $\boldsymbol{w}$ in (11) can be performed explicitly.

Instead we provide the following result, which shows that the step-direction of the EM-algorithm has an underlying relationship with the Newton method. In particular we show that, under suitable

regularity conditions, the direction of the EM-update, *i.e.* $\boldsymbol{w}_{k+1} - \boldsymbol{w}_k$, is the same, up to first order, as the direction of an approximate Newton method that uses $\mathcal{H}_2(\boldsymbol{w})$ in place of $\mathcal{H}(\boldsymbol{w})$.

**Theorem 1.** *Suppose we are given a Markov Decision Process with objective (1) and Markovian trajectory distribution (2). Consider the update of the parameter through Expectation Maximisation at the $k^{\text{th}}$ iteration of the algorithm,* i.e.

$$\boldsymbol{w}_{k+1} = \mathrm{argmax}_{\boldsymbol{w} \in \mathcal{W}} \ \mathcal{Q}(\boldsymbol{w}, \boldsymbol{w}_k).$$

*Provided that $\mathcal{Q}(\boldsymbol{w}, \boldsymbol{w}_k)$ is twice continuously differentiable in the first parameter we have that*

$$\boldsymbol{w}_{k+1} - \boldsymbol{w}_k = -\mathcal{H}_2^{-1}(\boldsymbol{w}_k)\nabla_{\boldsymbol{w}|\boldsymbol{w}=\boldsymbol{w}_k} U(\boldsymbol{w}) + \mathcal{O}(\|\boldsymbol{w}_{k+1} - \boldsymbol{w}_k\|^2). \tag{12}$$

*Additionally, in the case where the* log*-policy is quadratic the relation to the approximate Newton method is exact,* i.e. *the second term on the* r.h.s. *(12) is zero.*

*Proof.* The idea of the proof is simple and only involves performing a Taylor expansion of $\nabla_{\boldsymbol{w}}^{10} \mathcal{Q}(\boldsymbol{w}, \boldsymbol{w}_k)$. As $\mathcal{Q}$ is assumed to be twice continuously differentiable in the first component this Taylor expansion is possible and gives

$$\nabla_{\boldsymbol{w}}^{10} \mathcal{Q}(\boldsymbol{w}_{k+1}, \boldsymbol{w}_k) = \nabla_{\boldsymbol{w}}^{10} \mathcal{Q}(\boldsymbol{w}_k, \boldsymbol{w}_k) + \nabla_{\boldsymbol{w}}^{20} \mathcal{Q}(\boldsymbol{w}_k, \boldsymbol{w}_k)(\boldsymbol{w}_{k+1} - \boldsymbol{w}_k) + \mathcal{O}(\|\boldsymbol{w}_{k+1} - \boldsymbol{w}_k\|^2). \tag{13}$$

As $\boldsymbol{w}_{k+1} = \mathrm{argmax}_{\boldsymbol{w} \in \mathcal{W}} \ \mathcal{Q}(\boldsymbol{w}, \boldsymbol{w}_k)$ it follows that $\nabla_{\boldsymbol{w}}^{10} \mathcal{Q}(\boldsymbol{w}_{k+1}, \boldsymbol{w}_k) = 0$. This means that, upon ignoring higher order terms in $\boldsymbol{w}_{k+1} - \boldsymbol{w}_k$, the Taylor expansion (13) can be rewritten into the form

$$\boldsymbol{w}_{k+1} - \boldsymbol{w}_k = -\nabla_{\boldsymbol{w}}^{20} \mathcal{Q}(\boldsymbol{w}_k, \boldsymbol{w}_k)^{-1} \nabla_{\boldsymbol{w}}^{10} \mathcal{Q}(\boldsymbol{w}_k, \boldsymbol{w}_k). \tag{14}$$

The proof is completed by observing that $\nabla_{\boldsymbol{w}}^{10} \mathcal{Q}(\boldsymbol{w}_k, \boldsymbol{w}_k) = \nabla_{\boldsymbol{w}|\boldsymbol{w}=\boldsymbol{w}_k} U(\boldsymbol{w})$ and $\nabla_{\boldsymbol{w}}^{20} \mathcal{Q}(\boldsymbol{w}_k, \boldsymbol{w}_k) = \mathcal{H}_2(\boldsymbol{w}_k)$. The second statement follows because in the case where the $log$-policy is quadratic the higher order terms in the Taylor expansion vanish. $\square$

### 2.3 Summary

In this section we have provided a novel analysis of both natural gradient ascent and Expectation Maximisation when applied to the MDP framework. Previously, while both of these algorithms have proved popular methods for MDP optimisation, there has been little understanding of them in terms of their search-direction in the parameter space or their relation to the Newton method. Firstly, our analysis shows that the Fisher information matrix, which is used in natural gradient ascent, is similar to $\mathcal{H}_2(\boldsymbol{w})$ in (5) with the exception that the information about the reward structure of the problem is not contained in the Fisher information matrix, while such information is contained in $\mathcal{H}_2(\boldsymbol{w})$. Additionally we have shown that the step-direction of the EM-algorithm is, up to first order, an approximate Newton method that uses $\mathcal{H}_2(\boldsymbol{w})$ in place of $\mathcal{H}(\boldsymbol{w})$ and employs a constant step-size of one.

## 3  An Approximate Newton Method

A natural follow on from the analysis in section(2) is the consideration of using $\mathcal{M}(\boldsymbol{w}) = -\mathcal{H}_2^{-1}(\boldsymbol{w})$ in (3), a method we call the *full approximate Newton method* from this point onwards. In this section we show that this method has many desirable properties that make it an attractive alternative to other parametric policy search methods. Additionally, denoting the diagonal matrix formed from the diagonal elements of $\mathcal{H}_2(\boldsymbol{w})$ by $\mathcal{D}_2(\boldsymbol{w})$, we shall also consider the method that uses $\mathcal{M}(\boldsymbol{w}) = -\mathcal{D}_2^{-1}(\boldsymbol{w})$ in (3). We call this second method the *diagonal approximate Newton method*.

Recall that in (3) it is necessary that $\mathcal{M}(\boldsymbol{w})$ is positive-definite (in the Newton method this corresponds to requiring the Hessian to be negative-definite) to ensure an increase of the objective. In general the objective (1) is not concave, which means that the Hessian will not be negative-definite over the entire parameter space. In such cases the Newton method can actually lower the objective and this is an undesirable aspect of the Newton method. An attractive property of the approximate Hessian, $\mathcal{H}_2(\boldsymbol{w})$, is that it is always negative-definite when the policy is $\log$–concave in the policy parameters. This fact follows from the observation that in such cases $\mathcal{H}_2(\boldsymbol{w})$ is a non-negative mixture of negative-definite matrices, which again is negative-definite [9]. Additionally, the diagonal

terms of a negative-definite matrix are negative and so $\mathcal{D}_2(\boldsymbol{w})$ is also negative-definite when the controller is log-concave.

To motivate this result we now briefly consider some widely used policies that are either log-concave or blockwise log-concave. Firstly, consider the Gibb's policy, $\pi(\boldsymbol{a}|\boldsymbol{s};\boldsymbol{w}) \propto \exp \boldsymbol{w}^T \boldsymbol{\phi}(\boldsymbol{a},\boldsymbol{s})$, where $\boldsymbol{\phi}(\boldsymbol{a},\boldsymbol{s}) \in \mathbb{R}^{n_w}$ is a feature vector. This policy is widely used in discrete systems and is log-concave in $\boldsymbol{w}$, which can be seen from the fact that $\log \pi(\boldsymbol{a}|\boldsymbol{s};\boldsymbol{w})$ is the sum of a linear term and a negative *log-sum-exp* term, both of which are concave [9]. In systems with a continuous state-action space a common choice of controller is $\pi(\boldsymbol{a}|\boldsymbol{s};\boldsymbol{w}_{\text{mean}},\Sigma) = \mathcal{N}(\boldsymbol{a}|K\boldsymbol{\phi}(\boldsymbol{s})+\boldsymbol{m},\Sigma(\boldsymbol{s}))$, where $\boldsymbol{w}_{\text{mean}} = \{K,\boldsymbol{m}\}$ and $\boldsymbol{\phi}(\boldsymbol{s}) \in \mathbb{R}^{n_w}$ is a feature vector. The notation $\Sigma(\boldsymbol{s})$ is used because there are cases where is it beneficial to have state dependent noise in the controller. This controller is not jointly log-concave in $\boldsymbol{w}_{\text{mean}}$ and $\Sigma$, but it is blockwise log-concave in $\boldsymbol{w}_{\text{mean}}$ and $\Sigma^{-1}$. In terms of $\boldsymbol{w}_{\text{mean}}$ the log-policy is quadratic and the coefficient matrix of the quadratic term is negative-definite. In terms of $\Sigma^{-1}$ the log-policy consists of a linear term and a log-determinant term, both of which are concave.

In terms of evaluating the search direction it is clear from the forms of $\mathcal{D}_2(\boldsymbol{w})$ and $\mathcal{H}_2(\boldsymbol{w})$ that many of the pre-existing gradient evaluation techniques can be extended to the approximate Newton framework with little difficulty. In particular, gradient evaluation requires calculating the expectation of the derivative of the log-policy *w.r.t.* $p_\gamma(\boldsymbol{z};\boldsymbol{w})Q(\boldsymbol{z};\boldsymbol{w})$. In terms of inference the only additional calculation necessary to implement either the full or diagonal approximate Newton methods is the calculation of the expectation (*w.r.t.* to the same function) of the Hessian of the log-policy, or its diagonal terms. As an example in section(6.5) of the supplementary material we detail the extension of the recurrent state formulation of gradient evaluation in the average reward framework, see *e.g.* [31], to the approximate Newton method. We use this extension in the Tetris experiment that we consider in section(4). Given $n_s$ samples and $n_{\boldsymbol{w}}$ parameters the complexity of these extensions scale as $\mathcal{O}(n_s n_{\boldsymbol{w}})$ for the diagonal approximate Newton method, while it scales as $\mathcal{O}(n_s n_{\boldsymbol{w}}^2)$ for the full approximate Newton method.

An issue with the Newton method is the inversion of the Hessian matrix, which scales with $\mathcal{O}(n_{\boldsymbol{w}}^3)$ in the worst case. In the standard application of the Newton method this inversion has to be performed at each iteration and in large parameter systems this becomes prohibitively costly. In general $\mathcal{H}(\boldsymbol{w})$ will be dense and no computational savings will be possible when performing this matrix inversion. The same is not true, however, of the matrices $\mathcal{D}_2(\boldsymbol{w})$ and $\mathcal{H}_2(\boldsymbol{w})$. Firstly, as $\mathcal{D}_2(\boldsymbol{w})$ is a diagonal matrix it is trivial to invert. Secondly, there is an immediate source of sparsity that comes from taking the second derivative of the log-trajectory distribution in (7). This property ensures that any (product) sparsity over the control parameters in the log-trajectory distribution will correspond to sparsity in $\mathcal{H}_2(\boldsymbol{w})$. For example, in a *partially observable Markov Decision Processes* where the policy is modeled through a finite state controller, see *e.g.* [22], there are three functions to be optimised, namely the *initial belief distribution*, the *belief transition dynamics* and the *policy*. When the parameters of these three functions are independent $\mathcal{H}_2(\boldsymbol{w})$ will be block-diagonal (across the parameters of the three functions) and the matrix inversion can be performed more efficiently by inverting each of the block matrices individually. The reason that $\mathcal{H}(\boldsymbol{w})$ does not exhibit any such sparsity properties is due to the term $\mathcal{H}_1(\boldsymbol{w})$ in (5), which consists of the non-negative mixture of outer-product matrices. The vector in these outer-products is the derivative of the log-trajectory distribution and this typically produces a dense matrix.

A undesirable aspect of steepest gradient ascent is that its performance is affected by the choice of basis used to represent the parameter space. An important and desirable property of the Newton method is that it is invariant to non-singular linear (affine) transformations of the parameter space, see *e.g.* [9]. This means that given a non-singular linear (affine) mapping $\mathcal{T} \in \mathbb{R}^{n_w \times n_w}$, the Newton update of the objective $\tilde{U}(\boldsymbol{w}) = U(\mathcal{T}\boldsymbol{w})$ is related to the Newton update of the original objective through the same linear (affine) mapping, *i.e.* $\boldsymbol{v} + \Delta\boldsymbol{v}_{\text{nt}} = \mathcal{T}(\boldsymbol{w} + \Delta\boldsymbol{w}_{\text{nt}})$, where $\boldsymbol{v} = \mathcal{T}\boldsymbol{w}$ and $\Delta\boldsymbol{v}_{\text{nt}}$ and $\Delta\boldsymbol{w}_{\text{nt}}$ denote the respective Newton steps. In other words running the Newton method on $U(\boldsymbol{w})$ and $\tilde{U}(\mathcal{T}^{-1}\boldsymbol{w})$ will give identical results. An important point to note is that this desirable property is maintained when using $\mathcal{H}_2(\boldsymbol{w})$ in an approximate Newton method, while using $\mathcal{D}_2(\boldsymbol{w})$ results in a method that is invariant to rescaling of the parameters, *i.e.* where $\mathcal{T}$ is a diagonal matrix with non-zero elements along the diagonal. This can be seen by using the linearity of the expectation operator and a proof of this statement is provided in section(6.4) of the supplementary material.

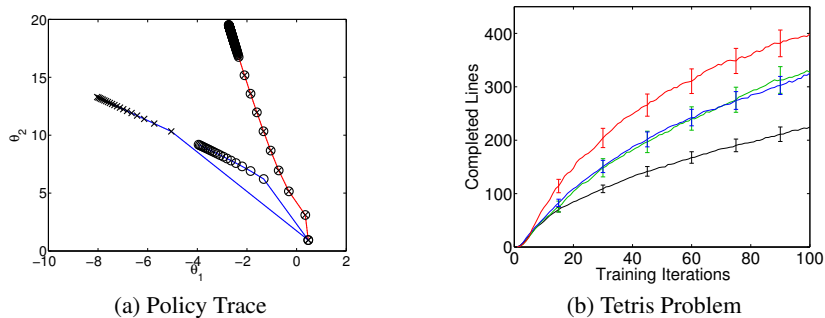

(a) Policy Trace　　　　　　　　　　　　　(b) Tetris Problem

Figure 1: (a) An empirical illustration of the affine invariance of the approximate Newton method, performed on the two state MDP of [16]. The plot shows the trace of the policy during training for the two different parameter spaces, where the results of the latter have been mapped back into the original parameter space for comparison. The plot shows the two steepest gradient ascent traces (blue cross and blue circle) and the two traces of the full approximate Newton method (red cross and red circle). (b) Results of the tetris problem for steepest gradient ascent (black), natural gradient ascent (green), the diagonal approximate Newton method (blue) and the full approximate Newton method (red).

## 4　Experiments

The first experiment we performed was an empirical illustration that the full approximate Newton method is invariant to linear transformations of the parameter space. We considered the simple two state example of [16] as it allows us to plot the trace of the policy during training, since the policy has only two parameters. The policy was trained using both steepest gradient ascent and the full approximate Newton method and in both the original and linearly transformed parameter space. The policy traces of the two algorithms are plotted in figure(1.a). As expected steepest gradient ascent is affected by such mappings, whilst the full approximate Newton method is invariant to them.

The second experiment was aimed at demonstrating the scalability of the full and diagonal approximate Newton methods to problems with a large state space. We considered the tetris domain, which is a popular computer game designed by Alexey Pajitnov in 1985. See [12] for more details. Firstly, we compared the performance of the full and diagonal approximate Newton methods to other parametric policy search methods. Tetris is typically played on a $20 \times 10$ grid, but due to computational costs we considered a $10 \times 10$ grid in the experiment. This results in a state space with roughly $7 \times 2^{100}$ states. We modelled the policy through a Gibb's distribution, where we considered a feature vector with the following features: the heights of each column, the difference in heights between adjacent columns, the maximum height and the number of 'holes'. Under this policy it is not possible to obtain the explicit maximum over $w$ in (11) and so a straightforward application of EM is not possible in this problem. We therefore compared the diagonal and full approximate Newton methods with steepest and natural gradient ascent. Due to reasons of space the exact implementation of the experiment is detailed in section(6.6) of the supplementary material. We ran 100 repetitions of the experiment, each consisting of 100 training iterations, and the mean and standard error of the results are given in figure(1.b). It can be seen that the full approximate Newton method outperforms all of the other methods, while the performance of the diagonal approximate Newton method is comparable to natural gradient ascent. We also ran several training runs of the full approximate Newton method on the full-sized $20 \times 10$ board and were able to obtain a score in the region of $14,000$ completed lines, which was obtained after roughly $40$ training iterations. An approximate dynamic programming based method has previously been applied to the Tetris domain in [7]. The same set of features were used and a score of roughly $4,500$ completed lines was obtained after around 6 training iterations, after which the solution then deteriorated.

In the third experiment we considered a finite horizon (controlled) linear dynamical system. This allowed the search-directions of the various algorithms to be computed exactly using [13] and removed any issues of approximate inference from the comparison. In particular we considered a 3-link rigid manipulator, linearized through feedback linearisation, see *e.g.* [17]. This system has a

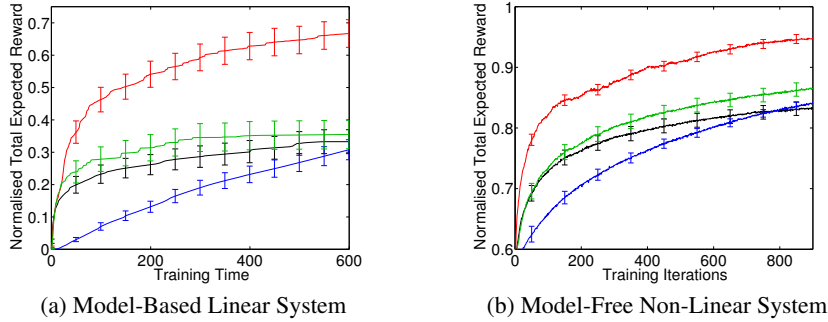

|  |  |
|---|---|
| (a) Model-Based Linear System | (b) Model-Free Non-Linear System |

Figure 2: (a) The normalised total expected reward plotted against training time, in seconds, for the 3-link rigid manipulator. The plot shows the results for steepest gradient ascent (black), EM (blue), natural gradient ascent (green) and the approximate Newton method (red), where the plot shows the mean and standard error of the results. (b) The normalised total expected reward plotted against training iterations for the synthetic non-linear system of [29]. The plot shows the results for EM (blue), steepest gradient ascent (black), natural gradient ascent (green) and the approximate Newton method (red), where the plot shows the mean and standard error of the results.

6-dimensional state space, 3-dimensional action space and a 22-dimensional parameter space. Further details of the system can be found in section(6.7) of the supplementary material. We ran the experiment 100 times and the mean and standard error of the results plotted in figure(2.a). In this experiment the approximate Newton method found substantially better solutions than either steepest gradient ascent, natural gradient ascent or Expectation Maximisation. The superiority of the results in comparison to either steepest or natural gradient ascent can be explained by the fact that $\mathcal{H}_2(\boldsymbol{w})$ gives a better estimate of the curvature of the objective function. Expectation Maximisation performed poorly in this experiment, exhibiting sub-linear convergence. Steepest gradient ascent performed $3684 \pm 314$ training iterations in this experiment which, in comparison to the $203 \pm 34$ and $310 \pm 40$ iterations of natural gradient ascent and the approximate Newton method respectively, illustrates the susceptibility of this method to poor scaling. In the final experiment we considered the synthetic non-linear system considered in [29]. Full details of the system and the experiment can be found in section(6.8) of the supplementary material. We ran the experiment 100 times and the mean and standard error of the results are plotted in figure(2.b). Again the approximate Newton method outperforms both steepest and natural gradient ascent. In this example only the mean parameters of the Gaussian controller are optimised, while the parameters of the noise are held fixed, which means that the log-policy is quadratic in the policy parameters. Hence, in this example the EM-algorithm is a particular (less general) version of the approximate Newton method, where a fixed step-size of one is used throughout. The marked difference in performance between the EM-algorithm and the approximate Newton method shows the benefit of being able to tune the step-size sequence. In this experiment we considered five different step-size sequences for the approximate Newton method and all of them obtained superior results than the EM-algorithm. In contrast only one of the seven step-size sequences considered for steepest and natural gradient ascent outperformed the EM-algorithm.

## 5 Conclusion

The contributions of this paper are twofold: Firstly we have given a novel analysis of Expectation Maximisation and natural gradient ascent when applied to the MDP framework, showing that both have close connections to an approximate Newton method; Secondly, prompted by this analysis we have considered the direct application of this approximate Newton method to the optimisation of MDPs, showing that it has numerous desirable properties that are not present in the naive application of the Newton method. In terms of empirical performance we have found the approximate Newton method to perform consistently well in comparison to EM and natural gradient ascent, highlighting its viability as an alternative to either of these methods. At present we have only considered *actor* type implementations of the approximate Newton method and the extension to *actor-critic* methods is a point of future research.

# References

[1] S. Amari. Natural Gradient Works Efficiently in Learning. *Neural Computation*, 10:251–276, 1998.

[2] M. Azar, V. Gómez, and H. Kappen. Dynamic policy programming with function approximation. *Journal of Machine Learning Research - Proceedings Track*, 15:119–127, 2011.

[3] J. Bagnell and J. Schneider. Covariant Policy Search. *IJCAI*, 18:1019–1024, 2003.

[4] J. Baxter and P. Bartlett. Infinite Horizon Policy Gradient Estimation. *Journal of Artificial Intelligence Research*, 15:319–350, 2001.

[5] D. P. Bertsekas. *Dynamic Programming and Optimal Control*. Athena Scientific, second edition, 2000.

[6] D. P. Bertsekas. Approximate Policy Iteration: A Survey and Some New Methods. Research report, Massachusetts Institute of Technology, 2010.

[7] D. P. Bertsekas and S. Ioffe. Temporal Differences-Based Policy Iteration and Applications in Neuro-Dynamic Programming. Research report, Massachusetts Institute of Technology, 1997.

[8] S. Bhatnagar, R. Sutton, M. Ghavamzadeh, and L. Mark. Natural Actor-Critic Algorithms. *Automatica*, 45:2471–2482, 2009.

[9] S. Boyd and L. Vandenberghe. *Convex Optimization*. Cambridge University Press, 2004.

[10] P. Dayan and G. E. Hinton. Using Expectation-Maximization for Reinforcement Learning. *Neural Computation*, 9:271–278, 1997.

[11] A. P. Dempster, N. M. Laird, and D. B. Rubin. Maximum Likelihood from Incomplete Data via the EM Algorithm. *Journal of the Royal Statistical Society. Series B (Methodological)*, 39(1):1–38, 1977.

[12] C. Fahey. Tetris AI, Computers Play Tetris `http://colinfahey.com/tetris/tetris_en.html`, 2003.

[13] T. Furmston and D. Barber. Efficient Inference for Markov Control Problems. *UAI*, 29:221–229, 2011.

[14] P. W. Glynn. Likelihood Ratio Gradient Estimation for Stochastic Systems. *Communications of the ACM*, 33:97–84, 1990.

[15] E. Greensmith, P. Bartlett, and J. Baxter. Variance Reduction Techniques For Gradient Based Estimates in Reinforcement Learning. *Journal of Machine Learning Research*, 5:1471–1530, 2004.

[16] S. Kakade. A Natural Policy Gradient. *NIPS*, 14:1531–1538, 2002.

[17] H. Khalil. *Nonlinear Systems*. Prentice Hall, 2001.

[18] J. Kober and J. Peters. Policy Search for Motor Primitives in Robotics. *Machine Learning*, 84(1-2):171–203, 2011.

[19] L. Kocsis and C. Szepesvári. Bandit Based Monte-Carlo Planning. *European Conference on Machine Learning (ECML)*, 17:282–293, 2006.

[20] V. R. Konda and J. N. Tsitsiklis. On Actor-Critic Algorithms. *SIAM J. Control Optim.*, 42(4):1143–1166, 2003.

[21] P. Marbach and J. Tsitsiklis. Simulation-Based Optimisation of Markov Reward Processes. *IEEE Transactions on Automatic Control*, 46(2):191–209, 2001.

[22] N. Meuleau, L. Peshkin, K. Kim, and L. Kaelbling. Learning Finite-State Controllers for Partially Observable Environments. *UAI*, 15:427–436, 1999.

[23] J. Nocedal and S. Wright. *Numerical Optimisation*. Springer, 2006.

[24] J. Peters and S. Schaal. Natural Actor-Critic. *Neurocomputing*, 71(7-9):1180–1190, 2008.

[25] K. Rawlik, Toussaint. M, and S. Vijayakumar. On Stochastic Optimal Control and Reinforcement Learning by Approximate Inference. *International Conference on Robotics Science and Systems*, 2012.

[26] S. Richter, D. Aberdeen, and J. Yu. Natural Actor-Critic for Road Traffic Optimisation. *NIPS*, 19:1169–1176, 2007.

[27] R. Sutton, D. McAllester, S. Singh, and Y. Mansour. Policy Gradient Methods for Reinforcement Learning with Function Approximation. *NIPS*, 13:1057–1063, 2000.

[28] M. Toussaint, S. Harmeling, and A. Storkey. Probabilistic Inference for Solving (PO)MDPs. Research Report EDI-INF-RR-0934, University of Edinburgh, School of Informatics, 2006.

[29] N. Vlassis, M. Toussaint, G. Kontes, and S. Piperidis. Learning Model-Free Robot Control by a Monte Carlo EM Algorithm. *Autonomous Robots*, 27(2):123–130, 2009.

[30] L. Weaver and N. Tao. The Optimal Reward Baseline for Gradient Based Reinforcement Learning. *UAI*, 17(29):538–545, 2001.

[31] R. Williams. Simple Statistical Gradient Following Algorithms for Connectionist Reinforcement Learning. *Machine Learning*, 8:229–256, 1992.

